# Learning Hybrid Models for Image Annotation with Partially Labeled Data

**Xuming He**
Department of Statistics
UCLA
hexm@stat.ucla.edu

**Richard S. Zemel**
Department of Computer Science
University of Toronto
zemel@cs.toronto.edu

## Abstract

Extensive labeled data for image annotation systems, which learn to assign class labels to image regions, is difficult to obtain. We explore a hybrid model framework for utilizing partially labeled data that integrates a generative topic model for image appearance with discriminative label prediction. We propose three alternative formulations for imposing a spatial smoothness prior on the image labels. Tests of the new models and some baseline approaches on three real image datasets demonstrate the effectiveness of incorporating the latent structure.

## 1 Introduction

Image annotation, or image labeling, in which the task is to label each pixel or region of an image with a class label, is becoming an increasingly popular problem in the machine learning and machine vision communities [7, 14]. State-of-the-art methods formulate image annotation as a structured prediction problem, and utilize methods such as Conditional Random Fields [8, 4], which output multiple values for each input item. These methods typically rely on fully labeled data for optimizing model parameters. It is widely acknowledged that consistently-labeled images are tedious and expensive to obtain, which limits the applicability of discriminative approaches. However, a large number of *partially-labeled* images, with a subset of regions labeled in an image, or only captions for images, are available (e.g., [12]). Learning labeling models with such data would help improve segmentation performance and relax the constraint of discriminative labeling methods.

A wide range of learning methods have been developed for using partially-labeled image data. One approach adopts a discriminative formulation, and treats the unlabeled regions as missing data [16], Others take a semi-supervised learning approach by viewing unlabeled image regions as unlabeled data. One class of these methods generalizes traditional semi-supervised learning to structured prediction tasks [1, 10]. However, the common assumption about the smoothness of the label distribution with respect to the input data may not be valid in image labeling, due to large intra-class variation of object appearance. Other semi-supervised methods adopt a *hybrid* approach, combining a generative model of the input data with a discriminative model for image labeling, in which the unlabeled data are used to regularize the learning of a discriminative model [6, 9]. Only relatively simple probabilistic models are considered in these approaches, without capturing the contextual information in images.

Our approach described in this paper extends the hybrid modeling strategy by incorporating a more flexible generative model for image data. In particular, we introduce a set of latent variables that capture image feature patterns in a hidden feature space, which are used to facilitate the labeling task. First, we extend the Latent Dirichlet Allocation model (LDA) [3] to include not only input features but also label information, capturing co-occurrences within and between image feature patterns and object classes in the data set. Unlike other topic models in image modeling [11, 18], our model integrates a generative model of image appearance and a discriminative model of region

labels. Second, the original LDA structure does not impose any spatial smoothness constraint to label prediction, yet incorporating such a spatial prior is important for scene segmentation. Previous approaches have introduced lateral connections between latent topic variables [17, 15]. However, this complicates the model learning, and as a latent representation of image data, the topic variables can be non-smooth over the image plane in general. In this paper, we model the spatial dependency of labels by two different structures: one introduces directed connections between each label variable and its neighboring topic variables, and the other incorporates lateral connections between label variables. We will investigate whether these structures effectively capture the spatial prior, and lead to accurate label predictions.

The remainder of this paper is organized as follows. The next section presents the base model, and two different extensions to handle label spatial dependencies. Section 3 and 4 define inference and learning procedures for these models. Section 5 describes experimental results, and in the final section we discuss the model limitations and future directions.

## 2 Model description

The structured prediction problem in image labeling can be formulated as follows. Let an image $\mathbf{x}$ be represented as a set of subregions $\{x_i\}_{i=1}^{N_x}$. The aim is to assign each $x_i$ a label $l_i$ from a categorical set $\mathcal{L}$. For instance, subregion $x_i$'s can be image patches or pixels, and $\mathcal{L}$ consists of object classes. Denote the set of labels for $\mathbf{x}$ as $\mathbf{l} = \{l_i\}_{i=1}^{N_x}$. A key issue in structured prediction concerns how to capture the interactions between labels in $\mathbf{l}$ given the input image.

**Model I.** We first introduce our base model for capturing individual patterns in image appearance and label space. Assume each subregion $x_i$ is represented by two features $(a_i, t_i)$, in which $a_i$ describes its appearance (including color, texture, etc.) in some appearance feature space $\mathcal{A}$ and $t_i$ is its position on the image plane $\mathcal{T}$. Our method focuses on the joint distribution of labels and subregion appearances given positions by modeling co-occurred patterns in the joint space of $\mathcal{L} \times \mathcal{A}$. We achieve this by extending the latent Dirichlet allocation model to include both label and appearance.

More specifically, we assume each observation pair $(a_i, l_i)$ in image $\mathbf{x}$ is generated from a mixture of $K$ hidden 'topic' components shared across the whole dataset, given the position information $t_i$. Following the LDA notation, the mixture proportion is denoted as $\theta$, which is image-specific and shares a common Dirichlet prior parameterized by $\alpha$. Also, $z_i$ is used as an indicator variable to specify from which hidden topic component the pair $(a_i, l_i)$ is generated. In addition, we use $\mathbf{a}$ to denote the appearance feature vector of each image, $\mathbf{z}$ for the indicator vector and $\mathbf{t}$ for the position vector. Our model defines a joint distribution of label variables $\mathbf{l}$ and appearance feature variables $\mathbf{a}$ given the position $\mathbf{t}$ as follows,

$$P_b(\mathbf{l}, \mathbf{a}|\mathbf{t}, \alpha) = \int_\theta [\prod_i \sum_{z_i} P(l_i|a_i, t_i, z_i)P(a_i|z_i)P(z_i|\theta)]P(\theta|\alpha)d\theta \qquad (1)$$

where $P(\theta|\alpha)$ is the Dirichlet distribution. We specify the appearance model $P(a_i|z_i)$ to be position invariant but the label predictor $P(l_i|a_i, t_i, z_i)$ depends on the position information. Those two components are formulated as follows, and the graphical representation of the model is shown in the left panel of Figure 1.

**(a) Label prediction module** $P(l_i|a_i, t_i, z_i)$. The label predictor $P(l_i|a_i, t_i, z_i)$ is modeled by a probabilistic classifier that takes $(a_i, t_i, z_i)$ as its input and produces a properly normalized distribution for $l_i$. Note that we represent $z_i$ in its '0-1' vector form when it is used as the classifier input. So if the dimension of $\mathcal{A}$ is $M$, then the input dimension of the classifier is $M + K + 2$. We use a MLP with one hidden layer in our experiments, although other strong classifiers are also feasible.

**(b) Image appearance module** $P(a_i|z_i)$. We follow the convention of topic models and model the topic conditional distributions of the image appearance using a multinomial distribution with parameters $\beta_{z_i}$. As the appearance features typically take on real values, we first apply k-means clustering to the image features $\{a_i\}$ to build a visual vocabulary $\mathcal{V}$. Thus a feature $a_i$ in the appearance space $\mathcal{A}$ can be represented as a visual word $v$, and we have $P(a_i = v|z_i = k) = \beta_{k,v}$.

While the topic prediction model in Equation 1 is able to capture regularly co-occurring patterns in the joint space of label and appearance, it ignores spatial priors on the label prediction. However,

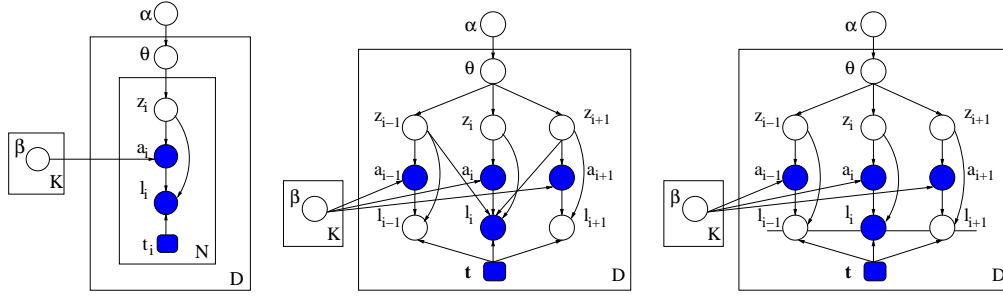

Figure 1: Left:A graphical representation of the base topic prediction model (Model I). Middle: Model II. Right: Model III. Circular nodes are random variables, and shaded nodes are observed. $N$ is the number of image features in each image, and $D$ denotes all the training data.

spatial priors, such as spatial smoothness, are crucial to labeling tasks, as neighboring labels are usually strongly correlated. To incorporate spatial information, we extend our base model in two different ways as follows.

**Model II.** We introduce a dependency between each label variable and its neighboring topic variables. In this model, each label value is predicted based on the summary information of topics within a neighborhood. More specifically, we change the label prediction model into the following form:

$$P(l_i|a_i, t_i, z_{N(i)}) = P(l_i|a_i, t_i, \sum_{j \in N(i)} w_j z_j), \tag{2}$$

where $N(i)$ is a predefined neighborhood for site $i$, and $w_j$ is the weight for the topic variable $z_j$. We set $w_j \propto \exp(-|t_i - t_j|/\sigma^2)$, and normalized to 1, i.e., $\sum_{j \in N(i)} w_j = 1$. The graphical representation is shown in the middle panel of Figure 1. This model variant can be viewed as an extension to the supervised LDA [2]. Here, however, rather than a single label applying to each input example instead there are multiple labels, one for each element of $\mathbf{x}$.

**Model III.** We add lateral connections between label variables to build a Conditional Random Field of labels. The joint label distribution given input image is defined as

$$P(\mathbf{l}|\mathbf{a}, \mathbf{t}, \alpha) = \frac{1}{Z} \exp\{\sum_{i,j \in N(i)} f(l_i, l_j) + \gamma \sum_i \log P_b(l_i|\mathbf{a}, \mathbf{t}, \alpha)\}, \tag{3}$$

where $Z$ is the partition function. The pairwise potential $f(l_i, l_j) = \sum_{a,b} u_{ab} \delta_{l_i,a} \delta_{l_j,b}$, and the unary potential is defined as log output of the base topic prediction model weighted by $\gamma$. Here $\delta$ is the Kronecker delta function. Note that $P_b(l_i|\mathbf{a}, \mathbf{t}, \alpha) = \sum_{z_i} P(l_i|a_i, t_i, z_i) P(z_i|\mathbf{a}, \mathbf{t})$. This model is shown in the right panel of Figure 1.

Note that the base model (Model I) obtains spatially smooth labels simply through the topics capturing location-dependent co-occurring appearance/label patterns, which tend to be nearby in image space. Model II explicitly predicts a region's label from the topics in its local neighborhood, so that neighboring labels share similar contexts defined by latent topics. In both of these models, the interaction between labels takes effect through the hidden input representation. The third model uses a conventional form of spatial dependency by directly incorporating local smoothing in the label field. While this structure may impose a stronger spatial prior than other two, it also requires more complicated learning methods.

## 3 Inference and Label Prediction

Given a new image $\mathbf{x} = \{\mathbf{a}, \mathbf{t}\}$ and our topic models, we predict its labeling based on the Maximum Posterior Marginals (MPM) criterion:

$$l_i^* = \arg \max_{l_i} P(l_i|\mathbf{a}, \mathbf{t}). \tag{4}$$

We consider the label inference procedure for three models separately as follows.

**Models I&II**: The marginal label distribution $P(l_i|\mathbf{a}, \mathbf{t})$ can be computed as:

$$P(l_i|\mathbf{a}, \mathbf{t}) = \sum_{z_{N(i)}} P(l_i|a_i, t_i, \sum_{j \in N(i)} w_j z_j) P(z_{N(i)}|\mathbf{a}, \mathbf{t}) \tag{5}$$

The summation here is difficult when $N(i)$ is large. However, it can be approximated as follows. Denote $v_i = \sum_{j \in N(i)} w_j z_j$ and $v_{i,q} = \sum_{j \in N(i)} w_j q(z_j)$, where $q(z_j) = \{P(z_j|\mathbf{a}, \mathbf{t})\}$ is the vector form of posterior distribution. Both $v_i$ and $v_{i,q}$ are in $[0,1]^K$. The marginal label distribution can be written as $P(l_i|\mathbf{a}, \mathbf{t}) = \langle P(l_i|a_i, t_i, v_i) \rangle_{P(z_{N(i)}|\mathbf{a}, \mathbf{t})}$. We take the first-order approximation of $P(l_i|a_i, t_i, v_i)$ around $v_{i,q}$ using Taylor expansion:

$$P(l_i|a_i, t_i, v_i) \approx P(l_i|a_i, t_i, v_{i,q}) + (v_i - v_{i,q})^T \cdot \nabla_{v_i} P(l_i|a_i, t_i, v_i)|_{v_{i,q}}. \tag{6}$$

Taking expectation on both sides of Equation 6 w.r.t. $P(z_{N(i)}|\mathbf{a}, \mathbf{t})$ (notice that $\langle v_i \rangle_{P(z_{N(i)}|\mathbf{a}, \mathbf{t})} = v_{i,q}$), we have the following approximation: $P(l_i|\mathbf{a}, \mathbf{t}) \approx \sum_{z_{N(i)}} P(l_i|a_i, t_i, \sum_{j \in N(i)} w_j q(z_j))$.

**Model III**: We first compute the unary potential of the CRF model from the base topic prediction model, i.e., $P_b(l_i|\mathbf{a}, \mathbf{t}) = \sum_{z_i} P(l_i|a_i, t_i, z_i) P(z_i|\mathbf{a}, \mathbf{t})$. Then the label marginals in Equation 4 are computed by applying loopy belief propagation to the conditional random field.

In both situations, we need the conditional distribution of the hidden topic variables $\mathbf{z}$ given observed data components to compute the label prediction. We take a Gibbs sampling approach by integrating out the Dirichlet variable $\theta$. From Equation 1, we can derive the posterior of each topic variable $z_i$ given other variables, which is required by Gibbs sampling:

$$P(z_i = k|\mathbf{z}_{-i}, a_i) \propto P(a_i|z_i)(\alpha_k + \sum_{m \in \mathcal{S} \setminus i} \delta_{z_m, k}) \tag{7}$$

where $\mathbf{z}_{-i}$ denotes all the topic variables in $\mathbf{z}$ except $z_i$, and $\mathcal{S}$ is the set of all sites. Given the samples of the topic variables, we estimate their posterior marginal distribution $P(z_i|\mathbf{a}, \mathbf{x})$ by simply computing their normalized histograms.

## 4    Learning with partially labeled data

Here we consider estimating the parameters of both extended models from a partially labeled image set $D = \{\mathbf{x}^n, \mathbf{l}^n\}$. For an image $\mathbf{x}^n$, its label $\mathbf{l}^n = (\mathbf{l}_o^n, \mathbf{l}_h^n)$ in which $\mathbf{l}_o^n$ denotes the observed labels, and $\mathbf{l}_h^n$ are missing. We also use $o$ to denote the set of labeled regions. As the three models are built with different components, we treat them separately.

**Models I&II.** We use the Maximum Likelihood criterion to estimate the model parameters. Let $\Theta$ be the parameter set of the model,

$$\Theta^* = \arg\max_{\Theta} \sum_n \log P(\mathbf{l}_o^n, \mathbf{a}^n | \mathbf{t}^n; \Theta) \tag{8}$$

We maximize the log data likelihood by Monte Carlo EM. The lower bound of the likelihood can be written as

$$Q = \sum_n \langle \sum_{i \in o} \log P(l_i^n|a_i^n, t_i^n, z_{N(i)}^n) + \sum_i \log P(a_i^n|z_i^n) + \log P(\mathbf{z}) \rangle_{P(\mathbf{z}^n|\mathbf{l}_o^n, \mathbf{a}^n)} \tag{9}$$

In the E step, the posterior distributions of the topic variables are estimated by a Gibbs sampling procedure similar to Equation 7. It uses the following conditional probability:

$$P(z_i = k|\mathbf{z}_{-i}, a_i, \mathbf{l}, \mathbf{t}) \propto \prod_{j \in N(i) \cap o} P(l_j|a_j, t_j, z_{N(j)}) P(a_i|z_i)(\alpha_k + \sum_{m \in \mathcal{S} \setminus i} \delta_{z_m, k}) \tag{10}$$

Note that any label variable is marginalized out if it is missing. In the M step, we update the model parameters by maximizing the lower bound $Q$. Denote the posterior distribution of $\mathbf{z}$ as $q(\cdot)$, the updating equation for parameters of the appearance module $P(a|z)$ can be derived from the stationary point of $Q$:

$$\beta_{k,v}^* \propto \sum_{n,i} q(z_i^n = k) \delta(a_i^n, v). \tag{11}$$

The classifier in the label prediction module is learned by maximizing the following log likelihood,

$$L_c = \sum_{n,i \in o} \langle \log P(l_i^n|a_i^n, t_i^n, \sum_{j \in N(i)} w_j z_j) \rangle_{q(z_{N(i)})} \approx \sum_{n,i \in o} \log P(l_i^n|a_i^n, t_i^n, \sum_{j \in N(i)} w_j q(z_j)). \tag{12}$$

where the approximation takes the same form as in Equation 6. We use a gradient ascent algorithm to update the classifier parameters. Note that we need to run only a few iterations at each M step, which reduces training time.

**Model III.** We estimate the parameters of Model III in two stages: (1). The parameters of the base topic prediction model are learned using the same procedure as in Models I&II. More specifically, we set $N(i) = i$ and estimate the parameters of the appearance module and label classifier based on Maximum Likelihood. (2). Given the base topic prediction model, we compute the marginal label probability $P_b(l_i|\mathbf{a}, \mathbf{t})$ and plug in the unary potential function in the CRF model (see Equation 3). We then estimate the parameters in the CRF by maximizing conditional pseudo-likelihood as follows:

$$L_p = \sum_n \sum_{i \in o} \left( \log \exp\{ \sum_{j \in N(i)} \sum_{a,b} u_{ab} \delta_{l_i^n,a} \delta_{l_j^n,b} + \gamma \log P_b(l_i^n|\mathbf{a}^n, \mathbf{t}^n) \} - \log Z_i^n \right). \tag{13}$$

where $Z_i^n = \sum_{l_i} \exp\{ \sum_{j \in N(i)} \sum_{a,b} u_{ab} \delta_{l_i,a} \delta_{l_j^n,b} + \gamma \log P_b(l_i|\mathbf{a}, \mathbf{t}) \}$ is the normalizing constant. As this cost function is convex, we use a simple gradient ascent method to optimize the conditional pseudo-likelihood.

## 5 Experimental evaluation

**Data sets and representation.** Our experiments are based on three image datasets. The first is a subset of the Microsoft Research Cambridge (MSRC) Image Database [14] as in [16]. This subset includes 240 images and 9 different label classes. The second set is the full MSRC image dataset, including 591 images and 21 object classes. The third set is a labeled subset of the Corel database as in [5] (referred therein as Corel-B). It includes 305 manually labeled images with 11 classes, focusing on animals and natural scenes.

We use the normalized cut segmentation algorithm [13] to build a super-pixel representation of the images, in which the segmentation algorithm is tuned to generate approximately 1000 segments for each image on average. We extract a set of basic image features, including color, edge and texture information, from each pixel site. For the color information, we transform the RGB values into CIE Lab* color space. The edge and texture are extracted by a set of filter-banks including a difference-of-Gaussian filter at 3 different scales, and quadrature pairs of oriented even- and odd-symmetric filters at 4 orientations and 3 scales.The color descriptor of a super-pixel is the average color over the pixels in that super-pixel. For edge and texture descriptors, we first discretize the edge/texture feature space by k-means, and use each cluster as a bin. Then we compute the normalized histograms of the features within a super-pixel as the edge/texture descriptor. In the experiments reported here, we used 20 bins for edge information and 50 bins for texture information. We also augment each feature by a SIFT descriptor extracted from a $30 \times 30$ image patch centered at the super-pixel. The image position of a super-pixel is the average position of its pixels. To compute the vocabulary of visual words in the topic model, we apply k-means to group the super-pixel descriptors into clusters. The cluster centers are used as visual words and each descriptor is encoded by its word index.

**Comparison methods.** We compare our approach directly with two baseline systems: a super-pixel-wise classifier and a basic CRF model. We also report the experimental results from [16], although they adopt a different data representation in their experiments (patches rather than super-pixels). The super-pixel-wise classifier is an MLP with one hidden layer, which predicts labels for each super-pixel independently. The MLP has 30 hidden units, a number chosen based on validation performance. In the basic CRF, the conditional distribution of the labels of an image is defined as:

$$P(\mathbf{l}|\mathbf{a}, \mathbf{t}) \propto \exp\{ \sum_{i,j} \sum_{u,v} \sigma_{u,v} \delta_{l_i,u} \delta_{l_j,v} + \gamma \sum_i h(l_i|a_i, t_i) \} \tag{14}$$

where $h(\cdot)$ is the log output from the super-pixel classifier. We train the CRF model by maximizing its conditional pseudo-likelihood, and label the image based on the marginal distribution of each label variable, computed by the loopy belief propagation algorithm.

**Performance on MSRC-9.** Following the setting in [16], we randomly split the dataset into training and testing sets with equal size, and use 10% training data as our validation set. In this experiment,

Table 1: A comparison of classification accuracy of the 3 variants of our model with other methods. The average classification accuracy is at the pixel level.

| Label | building | grass | tree | cow | sky | plane | face | car | bike | Total |
|---|---|---|---|---|---|---|---|---|---|---|
| S_Class | 61.2 | 93.2 | 71.3 | 57.0 | 92.9 | 37.5 | 69.0 | 56.0 | 54.1 | 74.2 |
| CRF | 69.8 | 94.4 | 82.1 | 73.3 | 94.2 | 62.0 | 80.5 | 80.1 | 78.6 | 83.5 |
| Model I | 64.8 | 93.0 | 76.6 | 72.0 | 93.5 | 65.1 | 74.4 | 61.3 | 77.7 | 79.7 |
| Model II | 79.2 | 94.1 | 81.4 | 80.2 | 93.5 | 72.4 | 86.3 | 69.5 | 86.2 | 85.5 |
| Model III | 78.1 | 92.5 | 85.4 | 86.7 | 94.6 | 77.9 | 83.5 | 74.7 | 88.3 | **86.7** |
| [16] | 73.6 | 91.1 | 82.1 | 73.6 | 95.7 | 78.3 | 89.5 | 84.5 | 81.4 | 84.9 |

we set the vocabulary size to 500, the number of hidden topics to 50, and each symmetric Dirichlet parameter $\alpha_k = 0.5$, based on validation performance. For Model II, we define the neighborhood of each site $i$ as a subset of sites that falls into a circular region centered at $i$ and with radius of $2\sigma$, where $\sigma$ is the fall-off rate of the weights. We set $\sigma$ to be 10 pixels, which is roughly $1/20$ of image size. The classifiers for label prediction have 15 hidden units. The appearance model for topics and the classifier are initialized randomly. In the learning procedure, the E step uses 500 samples to estimate the posterior distribution of topics. In the M step, we take 3 steps of gradient ascent learning of the classifiers per iteration.

The performance of our models is first evaluated on the dataset with all the labels available. We compare the performance of the three model variants to the super-pixel classifier (S_Class), and the CRF model. Table 1 shows the average classification accuracy rates of our model and the baselines for each class and in total, over 10 different random partitions of the dataset. We can see that Model I, which uses latent feature representations as additional inputs, achieves much better performance than the S_Class. Also, Model II and III improve the accuracy further by incorporating the label spatial priors. We notice that the lateral connections between label variables are more effective than integrating information from neighboring latent topic variables. This is also demonstrated by the good performance of the simple CRF.

**Learning with different amounts of label data.** In order to test the robustness of the latent feature representation, we evaluate our models using data with different amount of labeling information. We use an image dilation operator on the image regions labeled as 'void', and control the proportion of labeled data by varying the diameters of the dilation operator (see [16] for similar processing). Specifically, we use diameter values of 5, 10, 15, 20, 25, 30 and 35 to change the proportion of the labeled pixels to 62.9%, 52.1%, 44.1%, 36.4%, 30.5%, 24.9% and 20.3%, respectively. The original proportion is 71.9%. We report the average accuracies of 5 runs of training and testing with random equal partition of the dataset in Figure 2. The figure shows that the performance of all three models degrades with fewer labeled data, but the degradation is relatively gradual. When the proportion of labeled data decreases from 72% to 20%, the total loss in accuracy is less than 10%. This suggests that incorporating latent features makes our models more robust against missing labels than the previous work (cf. [16]). We also note that the performance of Model III is more robust than the other two variants, which may derive from stronger smoothing.

Table 2: A comparison of classification accuracy of our three model variants with other methods on the full MSRC dataset and Corel-B dataset.

| | S_Class | Model I | Model II | Model III | [14] | [5] |
|---|---|---|---|---|---|---|
| MSRC | 60.0 | 65.9 | 72.3 | 74.0 | 72.2 | - |
| Corel-B | 68.2 | 69.2 | 73.4 | 75.5 | - | 75.3 |

**Performance on other sets.** We further evaluate our models on two larger datasets to see whether they can scale up. The first dataset is the full version of the MSRC dataset, and we use the same training/testing partition as in [14]. The model setting is the same as in MSRC-9 except that we use a MLP with 20 hidden units for label prediction. The second is the Corel-B dataset, which is divided into 175 training images and 130 testing images randomly. We use the same setting of the models as in the experiments on the full MSRC set. Table 2 summarizes the classification accuracies of our models as well as some previous methods. For the full MSRC set, the two extended versions of our model achieve the similar performance as in [14], and we can see that the latent topic representation

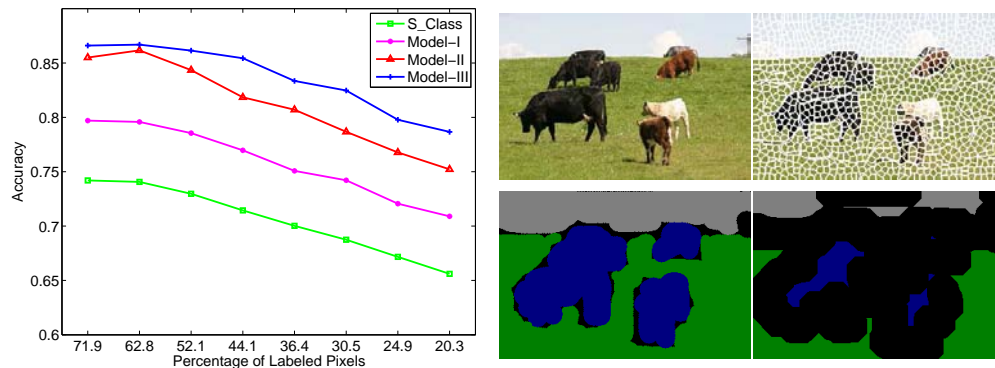

Figure 2: Left: Classification Accuracy with gradually decreasing proportion of labeled pixels. Right top: Examples of an image and its super-pixelization. Right bottom: Examples of original labeling and labeling after dilation (the ratio is 36.4).

provides useful cues. Also, our models have the same accuracy as reported in [5] on the Corel-B dataset, while we have a simpler label random field and use a smaller training set. It is interesting to note that the topics and spatial smoothness play less roles in the labeling performance on Corel-B. Figure 3 shows some examples of labeling results from both datasets. We can see that our models handle the extended regions better than those fine object structures, due to the tendency of (over)smoothing caused by super-pixelization and the two spatial dependency structures.

# 6   Discussion

In this paper, we presented a hybrid framework for image labeling, which combines a generative topic model with discriminative label prediction models. The generative model extends latent Dirichlet allocation to capture joint patterns in the label and appearance space of images. This latent representation of an image then provides an additional input to the label predictor. We also incorporated the spatial dependency into the model structure in two different ways, both imposing a prior of spatial smoothness for labeling on the image plane. The results of applying our methods to three different image datasets suggest that this integrated approach may extend to a variety of image databases with only partial labeling available. The labeling system consistently out-performs alternative approaches, such as a standard classifier and a standard CRF. Its performance also matches that of the state-of-the-art approaches, and is robust against different amount of missing labels.

Several avenues exist for future work. First, we would like to understand when the simple first-order approximation in inference for Model II holds, e.g., when the local curvature of the classifier with respect to its input is large. In addition, it is important to address model selection issues, such as the number of topics. We currently rely on the validation set, but more principled approaches are possible. A final issue concerns the reliance on visual words formed by clustering features in a complicated appearance space. Using a stronger appearance model may help us understand the role of different visual cues, as well as construct a more powerful generative model.

# References

[1] Yasemin Altun, David McAllester, and Mikhail Belkin. Maximum margin semi-supervised learning for structured variables. In *NIPS 18*, 2006.

[2] David Blei and Jon McAuliffe. Supervised topic models. In *NIPS 20*, 2008.

[3] David M. Blei, Andrew Y. Ng, and Michael I. Jordan. Latent Dirichlet allocation. *J. Mach. Learn. Res.*, 3:993–1022, 2003.

[4] Xuming He, Richard Zemel, and Miguel Carreira-Perpinan. Multiscale conditional random fields for image labelling. In *CVPR*, 2004.

[5] Xuming He, Richard S. Zemel, and Debajyoti Ray. Learning and incorporating top-down cues in image segmentation. In *ECCV*, 2006.

[6] Michael Kelm, Chris Pal, and Andrew McCallum. Combining generative and discriminative methods for pixel classification with multi-conditional learning. In *ICPR*, 2006.

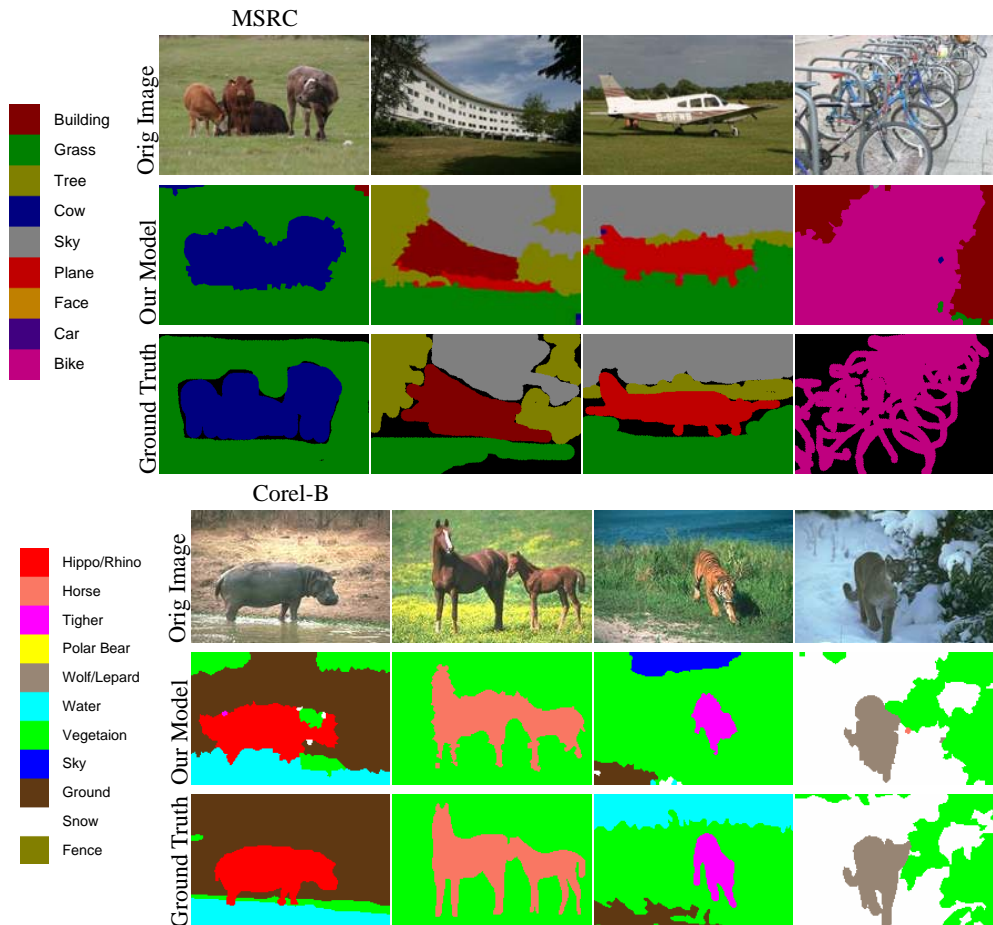

Figure 3: Some labeling results for the Corel-B (bottom panel) and MSRC-9 (top panel) datasets, based on the best performance of our models. The 'Void' region is annotated by color 'black'.

[7] Sanjiv Kumar and Martial Hebert. Discriminative random fields: A discriminative framework for contextual interaction in classification. In *ICCV*, 2003.

[8] John Lafferty, Andrew McCallum, and Fernando Pereira. Conditional random fields: Probabilistic models for segmenting and labeling sequence data. In *ICML*, pages 282–289, 2001.

[9] Julia A. Lasserre, Christopher M. Bishop, and Thomas P. Minka. Principled hybrids of generative and discriminative models. In *CVPR*, 2006.

[10] Chi-Hoon Lee, Shaojun Wang, Feng Jiao, Dale Schuurmans, and Russell Greiner. Learning to model spatial dependency: Semi-supervised discriminative random fields. In *NIPS 19*, 2007.

[11] Nicolas Loeff, Himanshu Arora, Alexander Sorokin, and David Forsyth. Efficient unsupervised learning for localization and detection in object categories. In *NIPS*, 2006.

[12] B. Russell, A. Torralba, K. Murphy, and W. Freeman. LabelMe: A database and web-based tool for image annotation. Technical report, MIT AI Lab Memo AIM-2005-025, 2005.

[13] J. Shi and J. Malik. Normalized cuts and image segmentation. *IEEE Trans. PAMI*, 2000.

[14] Jamie Shotton, John M. Winn, Carsten Rother, and Antonio Criminisi. Textonboost: Joint appearance, shape and context modeling for multi-class object recognition and segmentation. In *ECCV*, 2006.

[15] Jakob Verbeek and Bill Triggs. Region classification with markov field aspect models. In *CVPR*, 2007.

[16] Jakob Verbeek and Bill Triggs. Scene segmentation with CRFs learned from partially labeled images. In *NIPS 20*, 2008.

[17] Gang Wang, Ye Zhang, and Li Fei-Fei. Using dependent regions for object categorization in a generative framework. In *CVPR*, 2006.

[18] Xiaogang Wang and Eric Grimson. Spatial latent Dirichlet allocation. In *NIPS*, 2008.

